# Selective Attention for Handwritten Digit Recognition

**Ethem Alpaydın**
Department of Computer Engineering
Boğaziçi University
Istanbul, TR–80815 Turkey
alpaydin@boun.edu.tr

## Abstract

Completely parallel object recognition is NP-complete. Achieving a recognizer with feasible complexity requires a compromise between parallel and sequential processing where a system selectively focuses on parts of a given image, one after another. Successive fixations are generated to sample the image and these samples are processed and abstracted to generate a temporal context in which results are integrated over time. A computational model based on a partially recurrent feedforward network is proposed and made credible by testing on the real-world problem of recognition of handwritten digits with encouraging results.

## 1 INTRODUCTION

For all-parallel bottom-up recognition, allocating one separate unit for each possible feature combination, i.e., conjunctive encoding, implies combinatorial explosion. It has been shown that completely parallel, bottom-up visual object recognition is NP-complete (Tsotsos, 1990). By exchanging space with time, systems with much less complexity may be designed. For example, to phone someone at the press of a button, one needs $10^7$ buttons on the phone; the sequential alternative is to have 10 buttons on the phone and press one at a time, seven times.

We propose recognition based on selective attention where we analyze only a small part of the image in detail at each step, combining results in time. Noton and Stark's (1971) "scanpath" theory advocates that each object is internally represented as a feature-ring which is a temporal sequence of features extracted at each fixation and the positions or the motor commands for the eye movements in between. In this approach, there is an "eye" that looks at an image but which can really see only a small part of it. This part of the image that is examined in detail is the *fovea*. The

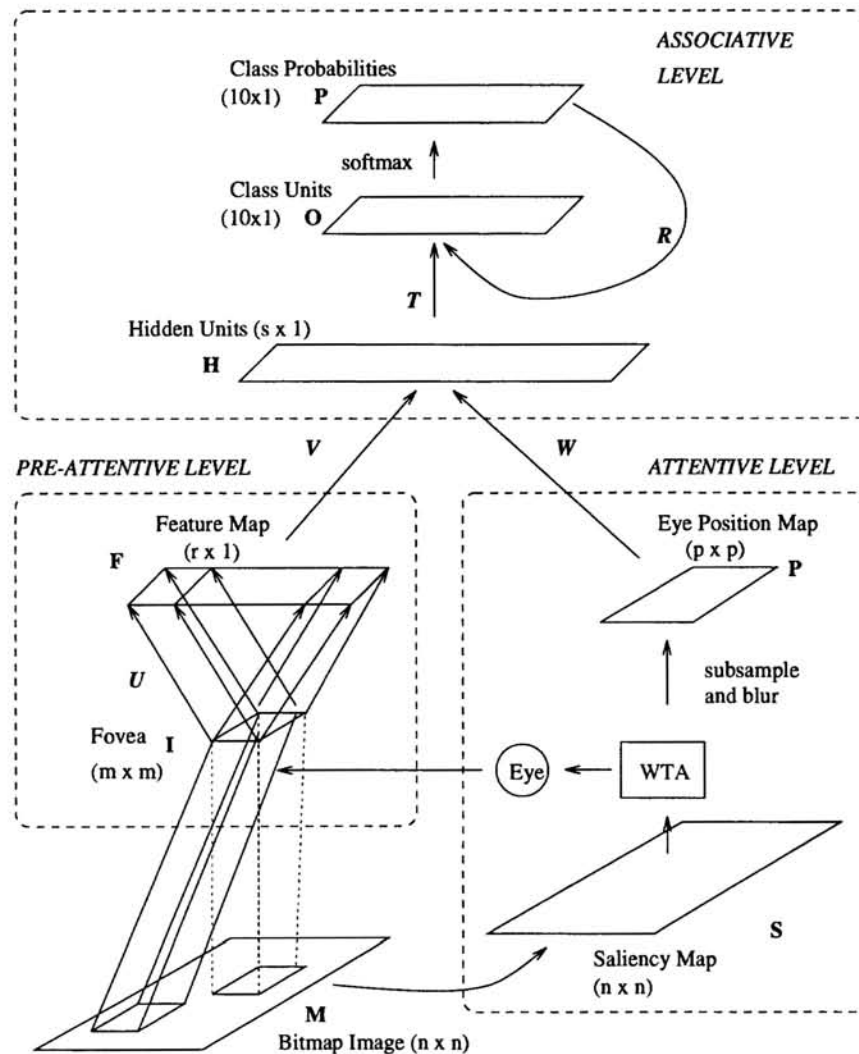

Figure 1: The block diagram of the implemented system.

fovea's content is examined by the *pre-attentive level* where basic feature extraction takes place. The features thus extracted are fed to an *associative* part together with the current eye position. If the accumulated information is not sufficient for recognition, the eye is moved to another part of the image, making a *saccade*. To minimize recognition time, the number of saccades should be minimized. This is done through defining a criterion of being "interesting" or saliency and by fixating only at the most interesting. Thus sucessive fixations are generated to sample the image and these samples are processed and abstracted to generate a temporal context in which results are integrated over time. There is a large amount of literature on selective attention in neuroscience and psychology; for reviews see respectively (Posner and Peterson, 1990) and (Treisman, 1988). The point stressed in this paper is that the approach is also useful in engineering.

## 2   AN EXAMPLE SYSTEM FOR OCR

The structure of the implemented system for recognition of handwritten digits is given in Fig. 1.

We have an $n \times n$ binary image in which the fovea is $m \times m$ with $m < n$. To minimize recognition time, the system should only attend to the parts of the image that carry discriminative information. We define a criterion of being "interesting" or saliency which is applied to all image locations in parallel to generate a *saliency map*, $S$. The saliency measure should be chosen to draw attention to parts that have the highest information content. Here, the saliency criterion is a low-pass filter which roughly counts the number of *on* pixels in the corresponding $m \times m$ region of the input image $M$. As the strokes in handwritten digits are mostly one or two pixels wide, a count of the *on* pixels is a good measure of the discontinuity (and thus information). It is also simple to compute:

$$S_{ij} = \sum_{k=i-\lfloor m/2 \rfloor}^{i+\lfloor m/2 \rfloor} \sum_{l=j-\lfloor m/2 \rfloor}^{j+\lfloor m/2 \rfloor} M_{kl} \mathcal{N}_2((i,j)^T, (\lfloor m/6 \rfloor)^2 * I), \quad i,j = 1 \ldots n$$

where $\mathcal{N}_2(\mu, \Sigma)$ is the bivariate normal with mean $\mu$ and the covariance $\Sigma$. Note that we want the convolution kernel to have effect up to $\lfloor m/2 \rfloor$ and also that the normal is zero after $\mu \pm 3\sigma$. In our simulations where $n$ is 16 and $m$ is 5 (typical for digit recognition), $\sigma \approx 1$. The location that is most salient is the position of the next fixation and as such defines the new center of the fovea. A location once attended to is no longer interesting; after each fixation, the saliency of all the locations that currently are in the scope of the fovea are set to 0 to inhibit another fixation there.

The attentive level thus controls the scope of the pre-attentive level. The maximum of the saliency map through a winner-take-all gives the eye position $(i^*, j^*)$ at fixation $t$.

$$(i^*(t), j^*(t)) = \arg\max_{i,j} S_{ij}$$

By thus following the salient regions, we get an input-dependent emergent sequence in time.

## Eye-Position Map

The *eye position map*, $P$, stores the position of the eye in the current fixation. It is $p \times p$. $p$ is chosen to be smaller than $n$ for dimensionality reduction for decreasing complexity and introducing an effect of regularization (giving invariance to small translations). When $p$ is a factor of $n$, computations are also simpler. We also blur the immediate neighbors for a smoother representation:

$$P(t) = \text{blur}(\text{subsample}(\text{winner-take-all}(S)))$$

## Pre-Attentive Level: Feature Extraction

The pre-attentive level extracts detailed features from the fovea to generate a *feature map*. This information and the current eye position is passed to the associative system for recognition. There is a trade-off between the fovea size and the number of saccades required for recognition: As the operation in the pre-attentive level is carried out in parallel, to minimize complexity the features extracted there should not be many and the fovea should not be large: Fovea is where the expensive computation takes place. On the other hand, the fovea should be large enough to extract discriminative features and thus complete recognition in a small amount of time. The features to be extracted can be learned through an supervised method when feedback is available.

The $m \times m$ region symmetrically around $(i^*, j^*)$ is extracted as the fovea $I$ and is fed to the feature extractors. The $r$ features extracted there are passed on to the associative level as the feature map, $F$. $r$ is typically 4 to 8. $U_g$ denote the weights of feature $g$ and $F_g$ is the value of feature $g$ that is found by convolving the fovea input with the feature weight vector ($f(\cdot)$ is the sigmoid function):

$$
\begin{aligned}
I_{ij}(t) &= M_{i^*(t)-\lfloor m/2 \rfloor + i, \, j^*(t)-\lfloor m/2 \rfloor + j}, \quad i,j = 1 \ldots m \\
F_{gij}(t) &= f\left( \sum_i \sum_j U_{gij} I_{ij}(t) \right), \quad g = 1 \ldots r
\end{aligned}
$$

**Associative Level: Classification**

At each fixation, the associative level is fed the feature map from the pre-attentive level and the eye position map from the attentive level. As a number of fixations may be necessary to recognize an image, the associative system should have a short-term memory able to accumulate inputs coming through time. Learning similarly should be through time. When used for classification, the class units are organized so as to compete and during recognition the activations of the class units evolve till one class gets sufficiently active and suppresses the others. When a training set is available, a temporal supervised method can be used to train the associative level. Note that there may be more than one scanpath for each object and learning one sequence for each object fails. We see it is a task of accumulating two types of information through time: the "what" (features extracted) and the "where" (eye position).

The fovea map, $F$, and the eye position map, $P$, are concatenated to make a $r + p \times p$ dimensional input that is fed to the associative level. Here we use an artificial neural network with one hidden layer of $s$ units. We have experimented with various architectures and noticed that recurrency at the output layer is the best. There are 10 output units.

$$
\begin{aligned}
H_h(t) &= f\left( \sum_g V_{hg} F_g(t) + \sum_a \sum_b W_{hab} P_{ab}(t) \right), \quad h = 1 \ldots s \\
O_c(t) &= \sum_h T_{ch} H_h + \sum_k R_{ck} P_k(t-1), \quad c = 1 \ldots 10 \\
P_c(t) &= \frac{\exp[O_c(t)]}{\sum_k \exp[O_k(t)]}
\end{aligned}
$$

where $P$ denotes the "softmax"ed output probabilities (Bridle, 1990) and $P(t-1)$ are the values in the preceding fixation (initially 0). We use the cross-entropy as the goodness measure:

$$
C = \sum_t \frac{1}{t} \sum_c D_k \log P_c(t), \quad t \geq 1
$$

$D_c$ is the required output for class $c$. Learning is gradient-ascent on this goodness measure. The fraction $1/t$ is to give more weight to initial fixations than later ones. Connections to the output units are updated as follows ($\eta$ is the learning factor):

$$\delta_c(t) = D_c - P_c(t) \quad \Delta T_{ch} = \frac{\eta}{t}\delta_c(t)H_h \quad R_{ck} = \frac{\eta}{t}\delta_c(t)P_k(t-1)$$

Note that we assume $\partial P_k(t-1)/\partial R_{ck} = 0$. For the connections to the hidden units we have:

$$\delta_h(t) = \sum_c \delta_c(t)T_{ch} \quad \Delta V_{hg}(t) = \frac{\eta}{t}\delta_h(t)F_g(t) \quad \Delta W_{hi}(t) = \frac{\eta}{t}\delta_h(t)P_i(t)$$

We can back-propagate one step more to train the feature extractors. Thus the update equations for the connections to feature units are:

$$\delta_g(t) = \sum_h \delta_h(t)V_{hg} \quad \Delta U_{gi}(t) = \frac{\eta}{t}\delta_g(t)I_i(t)$$

A series of fixations are made until one of the class units is sufficiently active: $\exists c, P_c > \theta$ (typically 0.99), or when the most salient point has a saliency less than a certain threshold (this condition is rarely met after the first few epochs). Then the computed changes are summed up and the updates are made like the exaple below:

$$\Delta T_{ch} = \sum_t \Delta T_{ch}(t)$$

Backpropagation through time where the recurrent connections are unfolded in time did not work well in this task because as explained before, for the same class, there is more than one scanpath. The above-mentioned approach is like real-time recurrent learning (Williams and Zipser, 1989) where the partial derivatives in the previous time step is 0, thus ignoring this temporal dependence.

## 3   RESULTS AND DISCUSSION

We have experimented with various parameter settings and finally chose the architecture given above: When input is $16 \times 16$ and there are 10 classes, the fovea is $5 \times 5$ with 8 features and there are 16 hidden units. There are 1,934 images for training, 946 for cross-validation and 943 for testing. Results are given in Table 1. ' It can be seen that by scanning less than half of the image, we get 80% generalization. Additional to the local high-resolution image provided by the fovea, a low-resolution image of the surrounding parafovea can be given to the associative level for better recognition. For example we low-pass filtered and undersampled the original image to get a $4 \times 4$ image which we fed to the class units additional to the attention-based hidden units. Success went up quite high and fewer fixations were necessary; compare rows 1 and 2 of the Table. The information provided by the $4 \times 4$ map is actually not much as can be seen from row 3 of the table where only that is given as input. Thus the idea is that when we have a coarse input, looking only at a quarter of the image in detail is sufficient to get 93% accuracy. Both features (what) and eye positions (where) are necessary for good recognition. When only one is used without the other, success is quite low as can be seen in rows 4 and 5. In the last row, we see the performance of a multi layer perceptron with 10 hidden units that does all-parallel recognition.

Beyond a certain network size, increasing the number of features do not help much. Decreasing $\theta$, the certainty threshold, decreases the number of fixations necessary

Table 1: Results of handwritten digit recognition with selective attention. Values given are average and standard deviation of 10 independent runs. See text for comments.

| METHOD | NO OF PARAMS | TEST SUCCESS | TRAINING EPOCHS | NO OF FIXATIONS |
|---|---|---|---|---|
| SA system | 878 | 79.7, 1.8 | 74.5, 17.1 | 6.5, 0.2 |
| SA+parafovea | 1,038 | 92.5, 0.8 | 54.2, 10.2 | 3.9, 0.3 |
| Only parafovea | 170 | 86.9, 0.2 | 52.3, 8.2 | 1.0, 0.0 |
| Only what info | 622 | 49.0, 21.0 | 66.6, 30.6 | 7.5, 0.1 |
| Only where info | 440 | 54.2, 1.4 | 92.9, 6.5 | 7.6, 0.0 |
| MLP, 10 hiddens | 2,680 | 95.1, 0.6 | 13.5, 4.1 | 1.0, 0.0 |

which we want, but decreases success too which we don't. Smaller foveas decrease the number of free parameters but decrease success and require a larger number of fixations. Similarly larger foveas decrease the number of fixations but increase complexity.

The simple low-pass filter used here as a saliency measure is the simplest measure. Previously it has been used by Fukushima and Imagawa (1993) for finding the next character, i.e., segmentation, and also by Olshausen et al. (1992) for translation invariance. More robust measures at the expense of more computations, are possible; see (Rimey and Brown, 1990; Milanese et al., 1993). Salient regions are those that are conspicious, i.e., different from their surrounding where there is a change in $X$ where $X$ can be brightness or color (edges), orientation (corners), time (motion), etc. It is also possible that top-down, task-dependent saliency measures be integrated to minimize further recognition time implying a remembered explicit sequence analogous to skilled motor behaviour (probably gained after many repetitions).

Here a partially recurrent network is used for temporal processing. Hidden Markov Models like used in speech recognition are another possibility (Rimey and Brown, 1990; Hacısalihzade et al., 1992). They are probabilistic finite automata which can be trained to classify sequences and one can have more than one model for an object.

It should be noted here that better approaches for the same problem exists (Le Cun et al., 1989). Here we advocate a computational model and make it plausible by testing it on a real-world problem. It is necessary for more complicated problems where an all-parallel approach would not work. For example Le Cun et al.'s model for the same type of inputs has 2,578 free parameters. Here there are

$$\underbrace{(m \times m + 1) \times r}_{U} + \underbrace{(r + p \times p + 1) \times s}_{V+W} + \underbrace{(s+1) \times 10}_{T} + \underbrace{10 \times 10}_{R}$$

free parameters which make 878 when $m = 5, r = 8, s = 16$. This is the main advantage of selective attention which is that the complexity of the system is heavily reduced at the expense of slower recognition, both in overt form of attention through foveation and in its covert form, for binding features — For this latter type of attention not discussed here, see (Ahmad, 1992). Also note that low-level feature extraction operations like carried out in the pre-attentive level are local convolutions

and are appropriate for parallel processing, e.g., on a SIMD machine. Higher-level operations require larger connectivity and are better carried out sequentially. Nature also seems to have taken this direction.

## Acknowledgements

This work is supported by Tübitak Grant EEEAG-143 and Boğaziçi University Research Funds 95HA108. Cenk Kaynak prepared the handwritten digit database based on the programs provided by NIST (Garris et al., 1994).

## References

S. Ahmad. (1992) VISIT: A Neural Model of Covert Visual Attention. In J. Moody, S. Hanson, R. Lippman (Eds.) *Advances in Neural Information Processing Systems 4*, 420–427. San Mateo, CA: Morgan Kaufmann.

J.S. Bridle. (1990) Probabilistic Interpretation of Feedforward Classification Network Outputs with Relationships to Statistical Pattern Recognition. In *Neurocomputing*, F. Fogelman-Soulié, J. Hérault, Eds. Springer, Berlin, 227–236.

K. Fukushima, T. Imagawa. (1993) Recognition and Segmentation of Connected Characters with Selective Attention, *Neural Networks*, **6**: 33–41.

M.D. Garris et al. (1994) NIST Form-Based Handprint Recognition System, NIS-TIR 5469, NIST Computer Systems Laboratory.

S.S. Hacısalihzade, L.W. Stark, J.S. Allen. (1992) Visual Perception and Sequences of Eye Movement Fixations: A Stochastic Modeling Approach, *IEEE SMC*, **22**, 474–481.

Y. Le Cun et al. (1991) Handwritten Digit Recognition with a Back-Propagation Network. In D.S. Touretzky (ed.) *Advances in Neural Information Processing Systems 2*, 396–404. San Mateo, CA: Morgan Kaufmann.

R. Milanese et al. (1994) Integration of Bottom-Up and Top-Down Cues for Visual Attention using Non-Linear Relaxation *IEEE Int'l Conf on CVPR*, Seattle, WA, USA.

D. Noton and L. Stark. (1971) Eye Movements and Visual Perception, *Scientific American*, **224**: 34–43.

B. Olshausen, C. Anderson, D. Van Essen. (1992) *A Neural Model of Visual Attention and Invariant Pattern Recognition*, CNS Memo 18, CalTech.

M.I. Posner, S.E. Petersen. (1990) The Attention System of the Human Brain, *Ann. Rev. Neurosci.*, **13**:25–42.

R.D. Rimey, C.M. Brown. (1990) *Selective Attention as Sequential Behaviour: Modelling Eye Movements with an Augmented Hidden Markov Model*, TR-327, Computer Science, Univ of Rochester.

A. Treisman. (1988) Features and Objects, *Quarterly Journ. of Exp. Psych.*, **40**: 201–237.

J.K. Tsotsos. (1990) Analyzing Vision at the Complexity Level, *Behav. and Brain Sci.* **13**: 423–469.

R.J. Williams, D. Zipser. (1989) A Learning Algorithm for Continually Running Fully Recurrent Neural Networks *Neural Computation*, **1**, 270–280.